# Polynomial Uniform Convergence of Relative Frequencies to Probabilities

Alberto Bertoni, Paola Campadelli,* Anna Morpurgo, Sandra Panizza
Dipartimento di Scienze dell'Informazione
Università degli Studi di Milano
via Comelico, 39 - 20135 Milano - Italy

## Abstract

We define the concept of *polynomial uniform convergence* of relative frequencies to probabilities in the *distribution-dependent* context. Let $X_n = \{0,1\}^n$, let $P_n$ be a probability distribution on $X_n$ and let $F_n \subset 2^{X_n}$ be a family of events. The family $\{\langle X_n, P_n, F_n \rangle\}_{n \geq 1}$ has the property of polynomial uniform convergence if the probability that the maximum difference (over $F_n$) between the relative frequency and the probability of an event exceed a given positive $\varepsilon$ be at most $\delta$ ($0 < \delta < 1$), when the sample on which the frequency is evaluated has size polynomial in $n, 1/\varepsilon, 1/\delta$. Given a $t$-sample $(x_1, \ldots, x_t)$, let $C_n^{(t)}(x_1, \ldots, x_t)$ be the Vapnik–Chervonenkis dimension of the family $\{\{x_1, \ldots, x_t\} \cap f \mid f \in F_n\}$ and $M(n,t)$ the expectation $\mathrm{E}(C_n^{(t)}/t)$. We show that $\{\langle X_n, P_n, F_n \rangle\}_{n \geq 1}$ has the property of polynomial uniform convergence iff there exists $\beta > 0$ such that $M(n,t) = O(n/t^\beta)$. Applications to distribution–dependent PAC learning are discussed.

## 1 INTRODUCTION

The probably approximately correct (PAC) learning model proposed by Valiant [Valiant, 1984] provides a complexity theoretical basis for learning from examples produced by an arbitrary distribution. As shown in [Blumer et al., 1989], a cen-

tral notion for distribution–free learnability is the Vapnik–Chervonenkis dimension, which allows obtaining estimations of the sample size adequate to learn at a given level of approximation and confidence. This combinatorial notion has been defined in [Vapnik & Chervonenkis, 1971] to study the problem of uniform convergence of relative frequencies of events to their corresponding probabilities in a distribution–free framework.

In this work we define the concept of *polynomial uniform convergence* of relative frequencies of events to probabilities in the *distribution–dependent* setting. More precisely, consider, for any $n$, a probability distribution on $\{0,1\}^n$ and a family of events $F_n \subseteq 2^{\{0,1\}^n}$: our request is that the probability that the maximum difference (over $F_n$) between the relative frequency and the probability of an event exceed a given arbitrarily small positive constant $\varepsilon$ be at most $\delta$ ($0 < \delta < 1$) when the sample on which we evaluate the relative frequencies has size polynomial in $n, 1/\varepsilon, 1/\delta$.

The main result we present here is a necessary and sufficient condition for polynomial uniform convergence in terms of "average information per example".

In section 2 we give preliminary notations and results; in section 3 we introduce the concept of polynomial uniform convergence in the distribution–dependent context and we state our main result, which we prove in section 4. Some applications to distribution–dependent PAC learning are discussed in section 5.

## 2   PRELIMINARY DEFINITIONS AND RESULTS

Let $X$ be a set of elementary events on which a probability measure $P$ is defined and let $F$ be a collection of boolean functions on $X$, i.e. functions $f : X \to \{0,1\}$. For $f \in F$ the set $f^{-1}(1)$ is said event, and $\mathcal{P}_f$ denotes its probability. A *t-sample* (or sample of size $t$) on $X$ is a sequence $\underline{x} = (x_1, \ldots, x_t)$, where $x_k \in X$ ($1 \le k \le t$). Let $X^{(t)}$ denote the space of $t$-samples and $P^{(t)}$ the probability distribution induced by $P$ on $X^{(t)}$, such that $P^{(t)}(x_1, \ldots, x_t) = P(x_1)P(x_2)\cdots P(x_t)$.

Given a $t$-sample $\underline{x}$ and a set $f \in F$, let $\nu_f^{(t)}(\underline{x})$ be the relative frequency of $f$ in the $t$-sample $\underline{x}$, i.e.

$$\nu_f^{(t)}(\underline{x}) = \frac{\sum_{i=1}^{t} f(x_i)}{t}.$$

Consider now the random variable $\Pi_F^{(t)} : X^{(t)} \to [0\ 1]$, defined over $\langle X^{(t)}, P^{(t)} \rangle$, where

$$\Pi_F^{(t)}(x_1, \ldots, x_t) = \sup_{f \in F} | \nu_f^{(t)}(x_1, \ldots, x_t) - \mathcal{P}_f | .$$

The relative frequencies of the events are said to converge to the probabilities uniformly over $F$ if, for every $\varepsilon > 0$, $\lim_{t \to \infty} P^{(t)}\{\underline{x} \mid \Pi_F^{(t)}(\underline{x}) > \varepsilon\} = 0$.

In order to study the problem of uniform convergence of the relative frequencies to the probabilities, the notion of index $\Delta_F(\underline{x})$ of a family $F$ with respect to a $t$-sample $\underline{x}$ has been introduced [Vapnik & Chervonenkis, 1971]. Fixed a $t$-sample $\underline{x} = (x_1, \ldots, x_t)$,

$$\Delta_F(\underline{x}) = \#\{f^{-1}(1) \cap \{x_1, \ldots, x_t\} | f \in F\}.$$

Obviously $\Delta_F(x_1, \ldots, x_t) \leq 2^t$; a set $\{x_1, \ldots, x_t\}$ is said *shattered* by $F$ iff $\Delta_F(x_1, \ldots, x_t) = 2^t$; the maximum $t$ such that there is a set $\{x_1, \ldots, x_t\}$ shattered by $F$ is said the *Vapnik-Chervonenkis dimension $d_F$* of $F$. The following result holds [Vapnik & Chervonenkis, 1971].

**Theorem 2.1** *For all distribution probabilities on $X$, the relative frequencies of the events converge (in probability) to their corresponding probabilities uniformly over $F$ iff $d_F < \infty$.*

We recall that the Vapnik–Chervonenkis dimension is a very useful notion in the distribution–independent PAC learning model [Blumer et al., 1989]. In the distribution–dependent framework, where the probability measure $P$ is fixed and known, let us consider the expectation $E[\log_2 \Delta_F(\underline{x})]$, called entropy $H_F(t)$ of the family $F$ in samples of size $t$; obviously $H_F(t)$ depends on the probability distribution $P$. The relevance of this notion is showed by the following result [Vapnik & Chervonenkis, 1971].

**Theorem 2.2** *A necessary and sufficient condition for the relative frequencies of the events in $F$ to converge uniformly over $F$ (in probability) to their corresponding probabilities is that*

$$\lim_{t \to \infty} \frac{H_F(t)}{t} = 0.$$

# 3   POLYNOMIAL UNIFORM CONVERGENCE

Consider the family $\{\langle X_n, P_n, F_n \rangle\}_{n \geq 1}$, where $X_n = \{0, 1\}^n$, $P_n$ is a probability distribution on $X_n$ and $F_n$ is a family of boolean functions on $X_n$.

Since $X_n$ is finite, the frequencies trivially converge uniformly to the probabilities; therefore we are interested in studying the problem of convergence with constraints on the sample size. To be more precise, we introduce the following definition.

**Definition 3.1** *Given the family $\{\langle X_n, P_n, F_n \rangle\}_{n \geq 1}$, the relative frequencies of the events in $F_n$ converge polynomially to their corresponding probabilities uniformly over $F_n$ iff there exists a polynomial $p(n, 1/\varepsilon, 1/\delta)$ such that*

$$\forall \varepsilon, \delta > 0 \, \forall n \, \left( t \geq p(n, 1/\varepsilon, 1/\delta) \Rightarrow P^{(t)}\{\underline{x} \mid \Pi_{F_n}^{(t)}(\underline{x}) > \varepsilon\} < \delta \right).$$

In this context $\varepsilon$ and $\delta$ are the approximation and confidence parameters, respectively.

The problem we consider now is to characterize the family $\{\langle X_n, P_n, F_n \rangle\}_{n \geq 1}$ such that the relative frequencies of events in $F_n$ converge polynomially to the probabilities. Let us introduce the random variable $C_n^{(t)} : X_n^{(t)} \to N$, defined as

$$C_n^{(t)}(x_1, \ldots, x_t) = \max\{\#A \mid A \subseteq \{x_1, \ldots, x_t\} \wedge A \text{ is shattered by } F_n\}.$$

In this notation it is understood that $C_n^{(t)}$ refers to $F_n$. The random variable $C_n^{(t)}$ and the index function $\Delta_{F_n}$ are related to one another; in fact, the following result can be easily proved.

**Lemma 3.1** $C_n^{(t)}(\underline{x}) \leq \log \Delta_{F_n}(\underline{x}) \leq C_n^{(t)}(\underline{x}) \log t.$

Let $M(n,t) = \mathrm{E}(\dfrac{C_n^{(t)}}{t})$ be the expectation of the random variable $\dfrac{C_n^{(t)}}{t}$. From Lemma 3.1 readily follows that

$$M(n,t) \leq \frac{H_{F_n}(t)}{t} \leq M(n,t) \log t;$$

therefore $M(n,t)$ is very close to $H_{F_n}(t)/t$, which can be interpreted as "average information for example" for samples of size $t$.

Our main result shows that $M(n,t)$ is a useful measure to verify whether $\{\langle X_n, P_n, F_n \rangle\}_{n \geq 1}$ satisfies the property of polynomial convergence, as shown by the following theorem.

**Theorem 3.1** *Given* $\{\langle X_n, P_n, F_n \rangle\}_{n \geq 1}$, *the following conditions are equivalent:*

**C1.** *The relative frequencies of events in $F_n$ converge polynomially to their corresponding probabilities.*

**C2.** *There exists $\beta > 0$ such that $M(n,t) = O(n/t^\beta)$.*

**C3.** *There exists a polynomial $\psi(n, 1/\varepsilon)$ such that*

$$\forall \varepsilon \, \forall n \, \left( t \geq \psi(n, 1/\varepsilon) \Rightarrow M(n,t) \leq \varepsilon \right).$$

*Proof.*

- **C2 $\Rightarrow$ C3** is readily veirfied. In fact, condition C2 says there exist $\alpha, \beta > 0$ such that $M(n,t) \leq \alpha n/t^\beta$; now, observing that $t \geq (\alpha n/\varepsilon)^{\frac{1}{\beta}}$ implies $\alpha n/t^\beta \leq \varepsilon$, condition C3 immediately follows.

- **C3 $\Rightarrow$ C2.** As stated by condition C3, there exist $a$, $b$, $c > 0$ such that if $t \geq an^b/\varepsilon^c$ then $M(n,t) \leq \varepsilon$. Solving the first inequality with respect to $\varepsilon$ gives, in the worst case, $\varepsilon = (an^b/t)^{\frac{1}{c}}$, and substituting for $\varepsilon$ in the second inequality yields $M(n,t) \leq (an^b/t)^{\frac{1}{c}} = a^{\frac{1}{c}} n^{\frac{b}{c}}/t^{\frac{1}{c}}$. If $\frac{b}{c} \leq 1$ we immediately obtain $M(n,t) \leq a^{\frac{1}{c}} n^{\frac{b}{c}}/t^{\frac{1}{c}} \leq a^{\frac{1}{c}} n/t^{\frac{1}{c}}$. Otherwise, if $\frac{b}{c} > 1$, since $M(n,t) \leq 1$, we have $M(n,t) \leq \min\{1, a^{\frac{1}{c}} n^{\frac{b}{c}}/t^{\frac{1}{c}}\} \leq \min\{1, (a^{\frac{1}{c}} n^{\frac{b}{c}}/t^{\frac{1}{c}})^{\frac{c}{b}}\} \leq a^{\frac{1}{b}} n/t^{\frac{1}{b}}$.    □

The proof of the equivalence between propositions C1 and C3 will be given in the next section.

# 4    PROOF OF THE MAIN THEOREM

First of all, we prove that condition C3 implies condition C1. The proof is based on the following lemma, which is obtained by minor modifications of [Vapnik & Chervonenkis, 1971 (Lemma 2, Theorem 4, and Lemma 4)].

**Lemma 4.1** *Given the family* $\{\langle X_n, P_n, F_n \rangle\}_{n \geq 1}$, *if* $\lim_{t \to \infty} \frac{H_{F_n}(t)}{t} = 0$ *then*

$$\forall \varepsilon \, \forall \delta \, \forall n \, \left( t \geq \frac{132 t_0}{\varepsilon^2 \delta} \Rightarrow P_n^{(t)}\{\underline{x} \mid \Pi_{F_n}^{(t)}(\underline{x}) > \varepsilon\} < \delta \right),$$

*where* $t_0$ *is such that* $H_{F_n}(t_0)/t_0 \leq \varepsilon^2/64$.

As a consequence, we can prove the following.

**Theorem 4.1** *Given* $\{\langle X_n, P_n, F_n \rangle\}_{n \geq 1}$, *if there exists a polynomial* $\psi(n, 1/\varepsilon)$ *such that*

$$\forall \varepsilon \, \forall n \, \left( t \geq \psi(n, 1/\varepsilon) \Rightarrow \frac{H_{F_n}(t)}{t} \leq \varepsilon \right),$$

*then the relative frequencies of events in* $F_n$ *converge polynomially to their probabilities.*

*Proof* (outline). It is sufficient to observe that if we choose $t_0 = \psi(n, 64/\varepsilon^2)$, by hypothesis it holds that $H_{F_n}(t_0)/t_0 \leq \varepsilon^2/64$; therefore, from Lemma 4.1, if

$$t \geq \frac{132 t_0}{\varepsilon^2 \delta} = \frac{132}{\varepsilon^2 \delta} \psi\left(n, \frac{64}{\varepsilon^2}\right),$$

then $P_n^{(t)}\{\underline{x} \mid \Pi_{F_n}^{(t)}(\underline{x}) > \varepsilon\} < \delta$.  □

An immediate consequence of Theorem 4.1 and of the relation $M(n, t) \leq H_{F_n}(t)/t \leq M(n, t) \log t$ is that condition C3 implies condition C1.

We now prove that condition C1 implies condition C3. For the sake of simplicity it is convenient to introduce the following notations:

$$a_n^{(t)} = \frac{C_n^{(t)}}{t} \qquad P_a(n, \varepsilon, t) = P_n^{(t)}\{\underline{x} \mid a_n^{(t)}(\underline{x}) \leq \varepsilon\}.$$

The following lemma, which relates the problem of polynomial uniform convergence of a family of events to the parameter $P_a(n, \varepsilon, t)$, will only be stated since it can be proved by minor modifications of Theorem 4 in [Vapnik & Chervonenkis, 1971].

**Lemma 4.2** *If* $t \geq 16/\varepsilon^2$ *then* $P_n^{(t)}\{\underline{x} \mid \Pi_{F_n}^{(t)}(\underline{x}) > \varepsilon\} \geq \frac{1}{4}(1 - P_a(n, 8\varepsilon, 2t))$.

A relevant property of $P_a(n, \varepsilon, t)$ is given by the following lemma.

**Lemma 4.3** $\forall \alpha \geq 1 \; P_a(n, \varepsilon/\alpha, \alpha t) \leq P_a^\alpha(n, \varepsilon, t)$.

*Proof.* Let $(\underline{x}_1, \ldots, \underline{x}_\alpha)$ be an $\alpha t$-sample obtained by the concatenation of $\alpha$ elements $\underline{x}_1, \ldots, \underline{x}_\alpha \in X^{(t)}$. It is easy to verify that $C_n^{(\alpha t)}(\underline{x}_1, \ldots, \underline{x}_\alpha) \geq \max_{i=1,\ldots,\alpha} C_n^{(t)}(\underline{x}_i)$. Therefore

$$P_n^{(\alpha t)}\{C_n^{(\alpha t)}(\underline{x}_1, \ldots, \underline{x}_\alpha) \leq k\} \leq P_n^{(\alpha t)}\{C_n^{(t)}(\underline{x}_1) \leq k \wedge \cdots \wedge C_n^{(t)}(\underline{x}_\alpha) \leq k\}.$$

By the independency of the events $C_n^{(t)}(\underline{x}_i) \leq k$ we obtain

$$P_n^{(\alpha t)}\{C_n^{(\alpha t)}(\underline{x}_1, \ldots, \underline{x}_\alpha) \leq k\} \leq \prod_{i=1}^{\alpha} P_n^{(t)}\{C_n^{(t)}(\underline{x}_i) \leq k\}.$$

Recalling that $a_n^{(t)} = C_n^{(t)}/t$ and substituting $k = \varepsilon t$, the thesis follows.   □

A relation between $P_a(n, \varepsilon, t)$ and the parameter $M(n,t)$, which we have introduced to characterize the polynomial uniform convergence of $\{\langle X_n, P_n, F_n\rangle\}_{n\geq 1}$, is shown in the following lemma.

**Lemma 4.4** *For every $\varepsilon$ ($0 < \varepsilon < 1/4$), if $M(n,t) > 2\sqrt{\varepsilon}$ then $P_a(n,\varepsilon,t) < 1/2$.*

*Proof.* For the sake of simplicity, let $m = M(n,l)$. If $m > \delta > 0$, we have

$$\delta < m = \int_0^1 x\,dP_a = \int_0^{\delta/2} x\,dP_a + \int_{\delta/2}^1 x\,dP_a$$

$$\leq \frac{\delta}{2}P_a(n,\frac{\delta}{2},l) + 1 - P_a(n,\frac{\delta}{2},l).$$

Since $0 < \delta < 1$, we obtain

$$P_a(n,\frac{\delta}{2},l) < \frac{1-\delta}{1-\delta/2} \leq 1 - \frac{\delta}{2}.$$

By applying Lemma 4.3 it is proved that, for every $\alpha \geq 1$,

$$P_a(n,\frac{\delta}{2\alpha},\alpha l) \leq \left(1 - \frac{\delta}{2}\right)^\alpha.$$

For $\alpha = \frac{2}{\delta}$ we obtain

$$P_a(n,\frac{\delta^2}{4},\frac{2l}{\delta}) < e^{-1} < \frac{1}{2}.$$

For $\varepsilon = \delta^2/4$ and $t = 2l/\delta$, the previous result implies that, if $M(n, t\sqrt{\varepsilon}) > 2\sqrt{\varepsilon}$, then $P_a(n, \varepsilon, t) < 1/2$.

It is easy to verify that $C_n^{(\alpha t)}(\underline{x}_1,\ldots,\underline{x}_\alpha) \leq \sum_{i=1}^\alpha C_n^{(t)}(\underline{x}_i)$ for every $\alpha \geq 1$. This implies $M(n,\alpha t) \leq M(n,t)$ for $\alpha \geq 1$, hence $M(n,t\sqrt{\varepsilon}) \geq M(n,t)$, from which the thesis follows.   □

**Theorem 4.2** *If for the family $\{\langle X_n, P_n, \mathcal{F}_n\rangle\}_{n\geq 1}$ the relative frequencies of events in $F_n$ converge polynomially to their probabilities, then there exists a polynomial $\psi(n,1/\varepsilon)$ such that*

$$\forall\varepsilon\,\forall n\,(t \geq \psi(n,1/\varepsilon) \Rightarrow M(n,t) \leq \varepsilon).$$

*Proof.* By contradiction. Let us suppose that $\{\langle X_n,P_n,F_n\rangle\}_{n\geq 1}$ polynomially converges and that for all polynomial functions $\psi(n,1/\varepsilon)$ there exist $\varepsilon, n, t$ such that $t \geq \psi(n,1/\varepsilon)$ and $M(n,t) > \varepsilon$.

Since $M(n,t)$ is a monotone, non–increasing function with respect to $t$ it follows that for every $\psi$ there exist $\varepsilon, n$ such that $M(n,\psi(n,1/\varepsilon)) > \varepsilon$. Considering the one-to-one corrispondence $T$ between polynomial functions defined by $T\psi(n,1/\varepsilon) = \varphi(n,4/\varepsilon^2)$, we can conclude that for any $\varphi$ there exist $\varepsilon, n$ such that $M(n,\varphi(n,1/\varepsilon)) > 2\sqrt{\varepsilon}$. From Lemma 4.4 it follows that

$$\forall\varphi\,\exists n\,\exists\varepsilon\,\left(P_a(n,\varepsilon,\varphi(n,\frac{1}{\varepsilon})) \leq \frac{1}{2}\right). \tag{1}$$

Since, by hypothesis, $\{\langle X_n, P_n, F_n \rangle\}_{n \geq 1}$ polynomially converges, fixed $\delta = 1/20$, there exists a polynomial $\phi$ such that

$$\forall \varepsilon \, \forall n \, \forall \phi \, \left( t \geq \phi(n, \frac{1}{\varepsilon}) \Rightarrow P_n^{(t)}\{\underline{x} \mid \Pi_{F_n}^{(t)}(\underline{x}) > \varepsilon\} < \frac{1}{20} \right)$$

From Lemma 4.2 we know that if $t \geq 16/\varepsilon^2$ then

$$P_n^{(t)}\{\underline{x} \mid \Pi_{F_n}^{(t)}(\underline{x}) > \varepsilon\} \geq \frac{1}{4}(1 - P_a(n, 8\varepsilon, 2t))$$

If $t \geq \max\{16/\varepsilon^2, \phi(n, 1/\varepsilon)\}$, then $\frac{1}{4}(1 - P_a(n, 8\varepsilon, 2t)) < \frac{1}{20}$, hence $P_a(n, 8\varepsilon, 2t) > \frac{4}{5}$.

Fixed a polynomial $\overline{p}(n, 1/\varepsilon)$ such that $2\overline{p}(n, 8/\varepsilon) \geq \max\{16/\varepsilon^2, \phi(n, 1/\varepsilon)\}$, we can conclude that

$$\forall \varepsilon \, \forall n \, \left( P_a(n, \varepsilon, \overline{p}(n, \frac{1}{\varepsilon})) > \frac{4}{5} \right). \tag{2}$$

From assertions (1) and (2) the contradiction $\frac{1}{2} < \frac{4}{5}$ can easily be derived.     □

An immediate consequence of Theorem 4.2 is that, in Theorem 3.1, condition C1 implies condition C3. Theorem 3.1 is thus proved.

## 5 DISTRIBUTION–DEPENDENT PAC LEARNING

In this section we briefly recall the notion of learnability in the distribution–dependent PAC model and we discuss some applications of the previous results. Given $\{\langle X_n, P_n, F_n \rangle\}_{n \geq 1}$, a labelled $t$-sample $S_f$ for $f \in F_n$ is a sequence $(\langle x_1, f(x_1) \rangle, \ldots, \langle x_t, f(x_t) \rangle)$, where $(x_1, \ldots, x_t)$ is a $t$-sample on $X_n$. We say that $f_1, f_2 \in F_n$ are $\varepsilon$-close with respect to $P_n$ iff $P_n\{x \mid f_1(x) \neq f_2(x)\} \leq \varepsilon$.

A learning algorithm $A$ for $\{\langle X_n, P_n, F_n \rangle\}_{n \geq 1}$ is an algorithm that, given in input $\varepsilon, \delta > 0$, a labelled $t$-sample $S_f$ with $f \in F_n$, outputs the representation of a function $g$ which, with probability $1 - \delta$, is $\varepsilon$-close to $f$. The family $\{\langle X_n, P_n, F_n \rangle\}_{n \geq 1}$ is said *polynomially learnable* iff there exists a learning algorithm $A$ working in time bounded by a polynomial $p(n, 1/\varepsilon, 1/\delta)$.

Bounds on the sample size necessary to learn at approximation $\varepsilon$ and confidence $1 - \delta$ have been given in terms of $\varepsilon$-covers [Benedek & Itai, 1988]; classes which are not learnable in the distribution–free model, but are learnable for some specific distribution, have been shown (e.g. $l$-terms DNF [Kucera et al., 1988]).

The following notion is expressed in terms of relative frequencies.

**Definition 5.1** *A quasi–consistent algorithm for the family $\{\langle X_n, P_n, F_n \rangle\}_{n \geq 1}$ is an algorithm that, given in input $\delta, \varepsilon > 0$ and a labelled $t$-sample $S_f$ with $f \in F_n$, outputs in time bounded by a polynomial $p(n, 1/\varepsilon, 1/\delta)$ the representation of a function $g \in F_n$ such that*

$$P_n^{(t)}\{\underline{x} \mid \nu_{f \oplus g}^{(t)}(\underline{x}) > \varepsilon\} < \delta$$

By Theorem 3.1 the following result can easily be derived.

**Theorem 5.1** *Given* $\{\langle X_n, P_n, F_n\rangle\}_{n\geq 1}$, *if there exists* $\beta > 0$ *such that* $M(n,t) = O(n/t^\beta)$ *and there exists a quasi-consistent algorithm for* $\{\langle X_n, P_n, F_n\rangle\}_{n\geq 1}$ *then* $\{\langle X_n, P_n, F_n\rangle\}_{n\geq 1}$ *is polynomially learnable.*

## 6  CONCLUSIONS AND OPEN PROBLEMS

We have characterized the property of polynomial uniform convergence of $\{\langle X_n, P_n, F_n\rangle\}_{n\geq 1}$ by means of the parameter $M(n,t)$. In particular we proved that $\{\langle X_n, P_n, \bar{F}_n\rangle\}_{n\geq 1}$ has the property of polynomial convergence iff there exists $\beta > 0$ such that $M(n,t) = O(n/t^\beta)$, but no attempt has been made to obtain better upper and lower bounds on the sample size in terms of $M(n,t)$.

With respect to the relation between polynomial uniform convergence and PAC learning in the distribution–dependent context, we have shown that if a family $\{\langle X_n, P_n, F_n\rangle\}_{n\geq 1}$ satisfies the property of polynomial uniform convergence then it can be PAC learned with a sample of size bounded by a polynomial function in $n$, $1/\varepsilon$, $1/\delta$.

It is an open problem whether the converse implication also holds.

### Acknowledgements

This research was supported by CNR, project Sistemi Informatici e Calcolo Parallelo.

## Footnotes

*Also at CNR, Istituto di Fisiologia dei Centri Nervosi, via Mario Bianco 9, 20131 Milano, Italy.

### References

G. Benedek, A. Itai. (1988) "Learnability by Fixed Distributions". *Proc. COLT'88*, 80-90.

A. Blumer, A. Ehrenfeucht, D. Haussler, K. Warmuth. (1989) "Learnability and the Vapnik–Chervonenkis Dimension". *J. ACM* **36**, 929-965.

L. Kucera, A. Marchetti–Spaccamela, M. Protasi. (1988) "On the Learnability of DNF Formulae". *Proc. XV Coll. on Automata, Languages, and Programming*, L.N.C.S. **317**, Springer Verlag.

L.G. Valiant. (1984) "A Theory of the Learnable". *Communications of the ACM* **27** (11), 1134-1142.

V.N. Vapnik, A.Ya. Chervonenkis. (1971) "On the uniform convergence of relative frequencies of events to their probabilities". *Theory of Prob. and its Appl.* **16** (2), 265-280.